# A Variational Bayesian Framework for Graphical Models

**Hagai Attias**
hagai@gatsby.ucl.ac.uk
Gatsby Unit, University College London
17 Queen Square
London WC1N 3AR, U.K.

## Abstract

This paper presents a novel practical framework for Bayesian model averaging and model selection in probabilistic graphical models. Our approach approximates full posterior distributions over model parameters and structures, as well as latent variables, in an analytical manner. These posteriors fall out of a free-form optimization procedure, which naturally incorporates conjugate priors. Unlike in large sample approximations, the posteriors are generally non-Gaussian and no Hessian needs to be computed. Predictive quantities are obtained analytically. The resulting algorithm generalizes the standard Expectation Maximization algorithm, and its convergence is guaranteed. We demonstrate that this approach can be applied to a large class of models in several domains, including mixture models and source separation.

## 1 Introduction

A standard method to learn a graphical model [1] from data is maximum likelihood (ML). Given a training dataset, ML estimates a single optimal value for the model parameters within a fixed graph structure. However, ML is well known for its tendency to overfit the data. Overfitting becomes more severe for complex models involving high-dimensional real-world data such as images, speech, and text. Another problem is that ML prefers complex models, since they have more parameters and fit the data better. Hence, ML cannot optimize model structure.

The Bayesian framework provides, in principle, a solution to these problems. Rather than focusing on a single model, a Bayesian considers a whole (finite or infinite) class of models. For each model, its posterior probability given the dataset is computed. Predictions for test data are made by averaging the predictions of all the individual models, weighted by their posteriors. Thus, the Bayesian framework avoids overfitting by integrating out the parameters. In addition, complex models are automatically penalized by being assigned a lower posterior probability, therefore optimal structures can be identified.

Unfortunately, computations in the Bayesian framework are intractable even for

very simple cases (e.g. factor analysis; see [2]). Most existing approximation methods fall into two classes [3]: Markov chain Monte Carlo methods and large sample methods (e.g., Laplace approximation). MCMC methods attempt to achieve exact results but typically require vast computational resources, and become impractical for complex models in high data dimensions. Large sample methods are tractable, but typically make a drastic approximation by modeling the posteriors over all parameters as Normal, even for parameters that are not positive definite (e.g., covariance matrices). In addition, they require the computation of the Hessian, which may become quite intensive.

In this paper I present *Variational Bayes* (VB), a practical framework for Bayesian computations in graphical models. VB draws together variational ideas from intractable latent variables models [8] and from Bayesian inference [4,5,9], which, in turn, draw on the work of [6]. This framework facilitates analytical calculations of posterior distributions over the hidden variables, parameters and structures. The posteriors fall out of a free-form optimization procedure which naturally incorporates conjugate priors, and emerge in standard forms, only one of which is Normal. They are computed via an iterative algorithm that is closely related to Expectation Maximization (EM) and whose convergence is guaranteed. No Hessian needs to be computed. In addition, averaging over models to compute predictive quantities can be performed analytically. Model selection is done using the posterior over structure; in particular, the BIC/MDL criteria emerge as a limiting case.

## 2   General Framework

We restrict our attention in this paper to directed acyclic graphs (DAGs, a.k.a. Bayesian networks). Let $Y = \{\mathbf{y}_1, ..., \mathbf{y}_N\}$ denote the visible (data) nodes, where $n = 1, ..., N$ runs over the data instances, and let $X = \{\mathbf{x}_1, ..., \mathbf{x}_N\}$ denote the hidden nodes. Let $\Theta$ denote the parameters, which are simply additional hidden nodes with their own distributions. A model with a fixed structure $m$ is fully defined by the joint distribution $p(Y, X, \Theta \mid m)$. In a DAG, this joint factorizes over the nodes, i.e. $p(Y, X \mid \Theta, m) = \prod_i p(u_i \mid \mathbf{pa}_i, \theta_i, m)$, where $u_i \in Y \cup X$, $\mathbf{pa}_i$ is the set of parents of $u_i$, and $\theta_i \in \Theta$ parametrize the edges directed toward $u_i$. In addition, we usually assume independent instances, $p(Y, X \mid \Theta, m) = \prod_n p(\mathbf{y}_n, \mathbf{x}_n \mid \Theta, m)$.

We shall also consider a set of structures $m \in M$, where $m$ controls the number of hidden nodes and the functional forms of the dependencies $p(u_i \mid \mathbf{pa}_i, \theta_i, m)$, including the range of values assumed by each node (e.g., the number of components in a mixture model). Associated with the set of structures is a structure prior $p(m)$.

**Marginal likelihood and posterior over parameters.** For a fixed structure $m$, we are interested in two quantities. The first is the *parameter posterior distribution* $p(\Theta \mid Y, m)$. The second is the *marginal likelihood* $p(Y \mid m)$, also known as the *evidence* assigned to structure $m$ by the data. In the following, the reference to $m$ is usually omitted but is always implied. Both quantities are obtained from the joint $p(Y, X, \Theta \mid m)$. For models with no hidden nodes the required computations can often be performed analytically. However, in the presence of hidden nodes, these quantities become computationally intractable. We shall approximate them using a variational approach as follows.

Consider the joint posterior $p(X, \Theta \mid Y)$ over hidden nodes and parameters. Since it is intractable, consider a *variational posterior* $q(X, \Theta \mid Y)$, which is restricted to the factorized form

$$q(X, \Theta \mid Y) = q(X \mid Y)q(\Theta \mid Y) \,, \tag{1}$$

where given the data, the parameters and hidden nodes are independent. This

restriction is the key: It makes $q$ approximate but tractable. Notice that we do not require complete factorization, as the parameters and hidden nodes may still be correlated amongst themselves.

We compute $q$ by optimizing a cost function $\mathcal{F}_m[q]$ defined by

$$\mathcal{F}_m[q] = \int d\Theta \, q(X)q(\Theta) \log \frac{p(Y, X, \Theta)}{q(X)q(\Theta)} \leq \log p(Y \mid m) \,, \qquad (2)$$

where the inequality holds for an arbitrary $q$ and follows from Jensen's inequality (see [6]); it becomes an equality when $q$ is the true posterior. Note that $q$ is always understood to include conditioning on $Y$ as in (1). Since $\mathcal{F}_m$ is bounded from above by the marginal likelihood, we can obtain the optimal posteriors by maximizing it w.r.t. $q$. This can be shown to be equivalent to minimizing the KL distance between $q$ and the true posterior. Thus, *optimizing $\mathcal{F}_m$ produces the best approximation to the true posterior within the space of distributions satisfying (1), as well as the tightest lower bound on the true marginal likelihood.*

**Penalizing complex models.** To see that the VB objective function $\mathcal{F}_m$ penalizes complexity, it is useful to rewrite it as

$$\mathcal{F}_m = \langle \log \frac{p(Y, X \mid \Theta)}{q(X)} \rangle_{X,\Theta} - KL[q(\Theta) \parallel p(\Theta)] \,, \qquad (3)$$

where the average in the first term on the r.h.s. is taken w.r.t. $q(X, \Theta)$. The first term corresponds to the (averaged) likelihood. The second term is the KL distance between the prior and posterior over the parameters. As the number of parameters increases, the KL distance follows and consequently reduces $\mathcal{F}_m$.

This penalized likelihood interpretation becomes transparent in the large sample limit $N \to \infty$, where the parameter posterior is sharply peaked about the most probable value $\Theta = \Theta_0$. It can then be shown that the KL penalty reduces to $(\mid \Theta_0 \mid /2) \log N$, which is linear in the number of parameters $\mid \Theta_0 \mid$ of structure $m$. $\mathcal{F}_m$ then corresponds precisely the Bayesian information criterion (BIC) and the minimum description length criterion (MDL) (see [3]). Thus, these popular model selection criteria follow as a limiting case of the VB framework.

**Free-form optimization and an EM-like algorithm.** Rather than assuming a specific parametric form for the posteriors, we let them fall out of free-form optimization of the VB objective function. This results in an iterative algorithm directly analogous to ordinary EM. In the E-step, we compute the posterior over the hidden nodes by solving $\delta \mathcal{F}_m / \delta q(X) = 0$ to get

$$q(X) \propto e^{\langle \log p(Y, X \mid \Theta) \rangle_\Theta} \,, \qquad (4)$$

where the average is taken w.r.t. $q(\Theta)$.

In the M-step, rather than the 'optimal' parameters, we compute the *posterior distribution over the parameters* by solving $\delta \mathcal{F}_m / \delta q(\Theta) = 0$ to get

$$q(\Theta) \propto e^{\langle \log p(Y, X \mid \Theta) \rangle_X} p(\Theta) \,, \qquad (5)$$

where the average is taken w.r.t. $q(X)$.

This is where the concept of conjugate priors becomes useful. Denoting the exponential term on the r.h.s. of (5) by $f(\Theta)$, we choose the prior $p(\Theta)$ from a family of distributions such that $q(\Theta) \propto f(\Theta)p(\Theta)$ belongs to that same family. $p(\Theta)$ is then said to be *conjugate* to $f(\Theta)$. This procedure allows us to select a prior from a fairly large family of distributions (which includes non-informative ones as limiting cases)

and thus not compromise generality, while facilitating mathematical simplicity and elegance. In particular, *learning in the VB framework simply amounts to updating the hyperparameters*, i.e., transforming the prior parameters to the posterior parameters. We point out that, while the use of conjugate priors is widespread in statistics, so far they could only be applied to models where all nodes were visible.

**Structure posterior.** To compute $q(m)$ we exploit Jensen's inequality once again to define a more general objective function, $\mathcal{F}[q] = \sum_{m \in M} q(m) \left[ \mathcal{F}_m + \log p(m)/q(m) \right] \leq \log p(Y)$, where now $q = q(X \mid m, Y) q(\Theta \mid m, Y) q(m \mid Y)$. After computing $\mathcal{F}_m$ for each $m \in M$, the structure posterior is obtained by free-form optimization of $\mathcal{F}$:

$$q(m) \propto e^{\mathcal{F}_m} p(m) . \tag{6}$$

Hence, prior assumptions about the likelihood of different structures, encoded by the prior $p(m)$, affect the selection of optimal model structures performed according to $q(m)$, as they should.

**Predictive quantities.** The ultimate goal of Bayesian inference is to estimate predictive quantities, such as a density or regression function. Generally, these quantities are computed by averaging over all models, weighting each model by its posterior. In the VB framework, exact model averaging is approximated by replacing the true posterior $p(\Theta \mid Y)$ by the variational $q(\Theta \mid Y)$. In density estimation, for example, the density assigned to a new data point $\mathbf{y}$ is given by $p(\mathbf{y} \mid Y) = \int d\Theta \, p(\mathbf{y} \mid \Theta) \, q(\Theta \mid Y)$.

In some situations (e.g. source separation), an estimate of hidden node values $\mathbf{x}$ from new data $\mathbf{y}$ may be required. The relevant quantity here is the conditional $p(\mathbf{x} \mid \mathbf{y}, Y)$, from which the most likely value of hidden nodes is extracted. VB approximates it by $p(\mathbf{x} \mid \mathbf{y}, Y) \propto \int d\Theta \, p(\mathbf{y}, \mathbf{x} \mid \Theta) \, q(\Theta \mid Y)$.

## 3    Variational Bayes Mixture Models

Mixture models have been investigated and analyzed extensively over many years. However, the well known problems of regularizing against likelihood divergences and of determining the required number of mixture components are still open. Whereas in theory the Bayesian approach provides a solution, no satisfactory practical algorithm has emerged from the application of involved sampling techniques (e.g., [7]) and approximation methods [3] to this problem. We now present the solution provided by VB.

We consider models of the form

$$p(\mathbf{y}_n \mid \Theta, m) = \sum_{s=1}^{m} p(\mathbf{y}_n \mid s_n = s, \Theta) \, p(s_n = s \mid \Theta) , \tag{7}$$

where $\mathbf{y}_n$ denotes the $n$th observed data vector, and $s_n$ denotes the hidden component that generated it. The components are labeled by $s = 1, ..., m$, with the structure parameter $m$ denoting the number of components. Whereas our approach can be applied to arbitrary models, for simplicity we consider here Normal component distributions, $p(\mathbf{y}_n \mid s_n = s, \Theta) = \mathcal{N}(\boldsymbol{\mu}_s, \boldsymbol{\Gamma}_\mathbf{s})$, where $\boldsymbol{\mu}_s$ is the mean and $\boldsymbol{\Gamma}_\mathbf{s}$ the precision (inverse covariance) matrix. The mixing proportions are $p(s_n = s \mid \Theta) = \pi_s$.

In hindsight, we use conjugate priors on the parameters $\Theta = \{\pi_s, \boldsymbol{\mu}_s, \boldsymbol{\Gamma}_s\}$. The mixing proportions are jointly Dirichlet, $p(\{\pi_s\}) = \mathcal{D}(\lambda^0)$, the means (conditioned on the precisions) are Normal, $p(\boldsymbol{\mu}_s \mid \boldsymbol{\Gamma}_s) = \mathcal{N}(\boldsymbol{\rho}^0, \beta^0 \boldsymbol{\Gamma}_s)$, and the precisions are Wishart, $p(\boldsymbol{\Gamma}_s) = \mathcal{W}(\nu^0, \boldsymbol{\Phi}^0)$. We find that the parameter posterior for a fixed $m$

factorizes into $q(\Theta) = q(\{\pi_s\}) \prod_s q(\mu_s, \Gamma_s)$. The posteriors are obtained by the following iterative algorithm, termed VB-MOG.

**E-step.** Compute the responsibilities for instance $n$ using (4):

$$\gamma_s^n \equiv q(s_n = s \mid \mathbf{y}_n) \propto \tilde{\pi}_s \ \tilde{\Gamma}_s^{1/2} \ e^{-(\mathbf{y}_n - \boldsymbol{\rho}_s)^T \tilde{\Gamma}_s (\mathbf{y}_n - \boldsymbol{\rho}_s)/2} \ e^{-d/2\beta_s} \ , \qquad (8)$$

noting that here $X = S$ and $q(S) = \prod_n q(s_n)$. This expression resembles the responsibilities in ordinary ML; the differences stem from integrating out the parameters. The special quantities in (8) are $\log \tilde{\pi}_s \equiv \langle \log \pi_s \rangle = \psi(\lambda_s) - \psi(\sum_{s'} \lambda_{s'})$, $\log \tilde{\Gamma}_s \equiv \langle \log \mid \Gamma_s \mid \rangle = \sum_{i=1}^{d} \psi((\nu_s + 1 - i)/2) - \log \mid \Phi_s \mid +d \log 2$, and $\tilde{\Gamma}_s \equiv \langle \Gamma_s \rangle = \nu_s \Phi_s^{-1}$, where $\psi(x) = d \log \Gamma(x)/dx$ is the digamma function, and the averages $\langle \cdot \rangle$ are taken w.r.t. $q(\Theta)$. The other parameters are described below.

**M-step.** Compute the parameter posterior in two stages. First, compute the quantities

$$\bar{\pi}_s = \frac{1}{N} \sum_{n=1}^{N} \gamma_s^n \ , \quad \bar{\mu}_s = \frac{1}{\bar{N}_s} \sum_{n=1}^{N} \gamma_s^n \, \mathbf{y}_n \ , \quad \bar{\Sigma}_s = \frac{1}{\bar{N}_s} \sum_{n=1}^{N} \gamma_s^n \, \mathbf{C}_s^n \ , \qquad (9)$$

where $\mathbf{C}_s^n = (\mathbf{y}_n - \bar{\mu}_s)(\mathbf{y}_n - \bar{\mu}_s)^T$ and $\bar{N}_s = N \bar{\pi}_s$. This stage is identical to the M-step in ordinary EM where it produces the new parameters. In VB, however, the quantities in (9) only help characterize the new parameter posteriors. These posteriors are functionally identical to the priors but have different parameter values. The mixing proportions are jointly Dirichlet, $q(\{\pi_s\}) = \mathcal{D}(\{\lambda_s\})$, the means are Normal, $q(\mu_s \mid \Gamma_s) = \mathcal{N}(\boldsymbol{\rho}_s, \beta_s \Gamma_s)$, and the precisions are Wishart, $p(\Gamma_s) = \mathcal{W}(\nu_s, \Phi_s)$. The posterior parameters are updated in the second stage, using the simple rules

$$\lambda_s = \bar{N}_s + \lambda^0 \ , \quad \boldsymbol{\rho}_s = (\bar{N}_s \bar{\mu}_s + \beta^0 \boldsymbol{\rho}^0)/(\bar{N}_s + \beta^0) \ , \quad \beta_s = \bar{N}_s + \beta^0 \ , \qquad (10)$$

$$\nu_s = \bar{N}_s + \nu^0 \ , \quad \Phi_s = \bar{N}_s \bar{\Sigma}_s + \bar{N}_s \beta^0 (\bar{\mu}_s - \boldsymbol{\rho}^0)(\bar{\mu}_s - \boldsymbol{\rho}^0)^T/(\bar{N}_s + \beta^0) + \Phi^0 \ .$$

The final values of the posterior parameters form the output of the VB-MOG. We remark that (a) Whereas no specific assumptions have been made about them, the parameter posteriors emerge in suitable, non-trivial (and generally non-Normal) functional forms. (b) The computational overhead of the VB-MOG compared to EM is minimal. (c) The covariance of the parameter posterior is $\mathcal{O}(1/N)$, and VB-MOG reduces to EM (regularized by the priors) as $N \to \infty$. (d) VB-MOG has no divergence problems. (e) Stability is guaranteed by the existence of an objective function. (f) Finally, the approximate marginal likelihood $\mathcal{F}_m$, required to optimize the number of components via (6), can also be obtained in closed form (omitted).

**Predictive Density.** Using our posteriors, we can integrate out the parameters and show that the density assigned by the model to a new data vector $\mathbf{y}$ is a mixture of Student-t distributions,

$$p(\mathbf{y} \mid Y) = \sum_{s=1}^{m} \bar{\pi}_s \, t_{\omega_s}(\mathbf{y} \mid \boldsymbol{\rho}_s, \Lambda_s) \ , \qquad (11)$$

where component $s$ has $\omega_s = \nu_s + 1 - d$ d.o.f., mean $\boldsymbol{\rho}_s$, covariance $\Lambda_s = ((\beta_s + 1)/\beta_s \omega_s)\Phi_s$, and proportion $\bar{\pi}_s = \lambda_s / \sum_{s'} \lambda_{s'}$. (11) reduces to a MOG as $N \to \infty$.

**Nonlinear Regression.** We may divide each data vector into input and output parts, $\mathbf{y} = (\mathbf{y}^i, \mathbf{y}^o)$, and use the model to estimate the regression function $\hat{\mathbf{y}}^o = f(\mathbf{y}^i)$ and error spheres. These may be extracted from the conditional $p(\mathbf{y}^o \mid \mathbf{y}^i, Y) = \sum_{s=1}^{m} w_s \, t_{\omega_s'}(\mathbf{y}^o \mid \boldsymbol{\rho}_s', \Lambda_s')$, which also turns out to be a mixture of Student-t distributions, with means $\boldsymbol{\rho}_s'$ being linear, and covariances $\Lambda_s'$ and mixing proportions $w_s$ nonlinear, in $\mathbf{y}^i$, and given in terms of the posterior parameters.

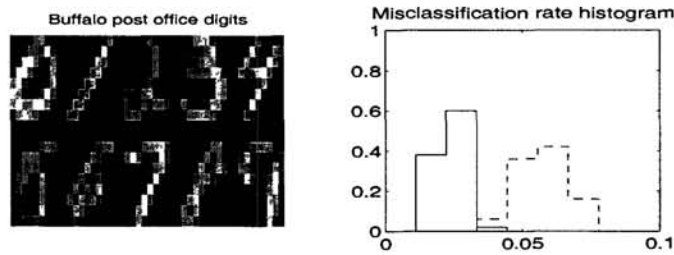

Figure 1: VB-MOG applied to handwritten digit recognition.

VB-MOG was applied to the Boston housing dataset (UCI machine learning repository), where 13 inputs are used to predict the single output, a house's price. 100 random divisions of the $N = 506$ dataset into 481 training and 25 test points were used, resulting in an average MSE of 11.9. Whereas ours is not a discriminative method, it was nevertheless competitive with Breiman's (1994) bagging technique using regression trees (MSE=11.7). For comparison, EM achieved MSE=14.6.

**Classification.** Here, a separate parameter posterior is computed for each class $c$ from a training dataset $Y^c$. Test data vector $\mathbf{y}$ is then classified according to the conditional $p(c \mid \mathbf{y}, \{Y^c\})$, which has a form identical to (11) (with $c$-dependent parameters) multiplied by the relative size of $Y^c$.

VB-MOG was applied to the Buffalo post office dataset, which contains 1100 examples for each digit $0 - 9$. Each digit is a gray-level $8 \times 8$ pixel array (see examples in Fig. 1 (left)). We used 10 random 500-digit batches for training, and a separate batch of 200 for testing. An average misclassification rate of .018 was obtained using $m = 30$ components; EM achieved .025. The misclassification histograms (VB=solid, EM=dashed) are shown in Fig. 1 (right).

## 4   VB and Intractable Models: a Blind Separation Example

The discussion so far assumed that a free-form optimization of the VB objective function is feasible. Unfortunately, for many interesting models, in particular models where ordinary ML is intractable, this is not the case. For such models, we modify the VB procedure as follows: (a) Specify a *parametric functional form* for the posterior over the hidden nodes $q(X)$, and optimize w.r.t. its parameters, in the spirit of [8]. (b) Let the parameter posterior $q(\Theta)$ fall out of free-form optimization, as before.

We illustrate this approach in the context of the blind source separation (BSS) problem (see, e.g., [1]). This problem is described by $\mathbf{y}_n = \mathbf{H}\mathbf{x}_n + \mathbf{u}_n$, where $\mathbf{x}_n$ is an unobserved $m$-dim source vector at instance $n$, $\mathbf{H}$ is an unknown mixing matrix, and the noise $\mathbf{u}_n$ is Normally distributed with an unknown precision $\lambda\mathbf{I}$. The task is to construct a source estimate $\hat{\mathbf{x}}_n$ from the observed $d$-dim data $\mathbf{y}$. The sources are independent and non-Normally distributed. Here we assume the high-kurtosis distribution $p(x_i^n) \propto \cosh^{-2}(x_i^n/2)$, which is appropriate for modeling speech sources. One important but heretofore unresolved problem in BSS is determining the number $m$ of sources from data. Another is to avoid overfitting the mixing matrix. Both problems, typical to ML algorithms, can be remedied using VB.

It is the non-Normal nature of the sources that renders the source posterior $p(X \mid Y)$ intractable even before a Bayesian treatment. We use a Normal variational posterior $q(X) = \prod_n \mathcal{N}(\mathbf{x}_n \mid \boldsymbol{\rho}_n, \boldsymbol{\Gamma}_n)$ with instance-dependent mean and precision. The mixing matrix posterior $q(\mathbf{H})$ then emerges as Normal. For simplicity, $\lambda$ is optimized rather than integrated out. The resulting VB-BSS algorithm runs as follows:

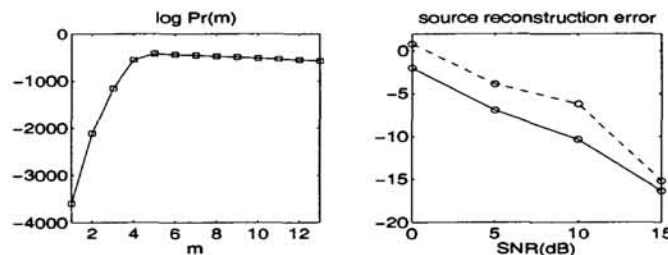

Figure 2: Application of VB to blind source separation algorithm (see text).

**E-step.** Optimize the variational mean $\boldsymbol{\rho}_n$ by iterating to convergence, for each $n$, the fixed-point equation $\lambda \bar{\mathbf{H}}^T(\mathbf{y}_n - \bar{\mathbf{H}}\boldsymbol{\rho}_n) - \tanh\boldsymbol{\rho}_n/2 = \mathbf{C}^{-1}\boldsymbol{\rho}_n$, where $\mathbf{C}$ is the source covariance conditioned on the data. The variational precision matrix turns out to be $n$-independent: $\boldsymbol{\Gamma}_n = \bar{\mathbf{A}}^T\Lambda\bar{\mathbf{A}} + \mathbf{I}/2 + \mathbf{C}^{-1}$.
**M-step.** Update the mean and precision of the posterior $q(\mathbf{H})$ (rules omitted).

This algorithm was applied to 11-dim data generated by linearly mixing 5 100msec-long speech and music signals obtained from commercial CDs. Gaussian noise were added at different SNR levels. A uniform structure prior $p(m) = 1/K$ for $m \leq K$ was used. The resulting posterior over the number of sources (Fig. 2 (left)) is peaked at the correct value $m = 5$. The sources were then reconstructed from test data via $p(\mathbf{x} \mid \mathbf{y}, Y)$. The log reconstruction error is plotted vs. SNR in Fig. 2 (right, solid). The ML error (which includes no model averaging) is also shown (dashed) and is larger, reflecting overfitting.

## 5 Conclusion

The VB framework is applicable to a large class of graphical models. In fact, it may be integrated with the junction tree algorithm to produce general inference engines with minimal overhead compared to ML ones. Dirichlet, Normal and Wishart posteriors are not special to models treated here but emerge as a general feature. Current research efforts include applications to multinomial models and to learning the structure of complex dynamic probabilistic networks.

### Acknowledgements
I thank Matt Beal, Peter Dayan, David Mackay, Carl Rasmussen, and especially Zoubin Ghahramani, for important discussions.

## Footnotes

[1]We use the term 'model' to refer collectively to parameters and structure.

### References
[1] Attias, H. (1999). Independent Factor Analysis. *Neural Computation* **11**, 803-851.
[2] Bishop, C.M. (1999). Variational Principal Component Analysis. *Proc. 9th ICANN*.
[3] Chickering, D.M. & Heckerman, D. (1997). Efficient approximations for the marginal likelihood of Bayesian networks with hidden variables. *Machine Learning* **29**, 181-212.
[4] Hinton, G.E. & Van Camp, D. (1993). Keeping neural networks simple by minimizing the description length of the weights. *Proc. 6th COLT*, 5-13.
[5] Jaakkola, T. & Jordan, M.I. (1997). Bayesian logistic regression: A variational approach. *Statistics and Artificial Intelligence* **6** (Smyth, P. & Madigan, D., Eds).
[6] Neal, R.M. & Hinton, G.E. (1998). A view of the EM algorithm that justifies incremental, sparse, and other variants. *Learning in Graphical Models*, 355-368 (Jordan, M.I., Ed). Kluwer Academic Press, Norwell, MA.
[7] Richardson, S. & Green, P.J. (1997). On Bayesian analysis of mixtures with an unknown number of components. *Journal of the Royal Statistical Society B*, **59**, 731-792.
[8] Saul, L.K., Jaakkola, T., & Jordan, M.I. (1996). Mean field theory of sigmoid belief networks. *Journal of Artificial Intelligence Research* **4**, 61-76.
[9] Waterhouse, S., Mackay, D., & Robinson, T. (1996). Bayesian methods for mixture of experts. *NIPS-8* (Touretzky, D.S. et al, Eds). MIT Press.